# Overfitting in Neural Nets: Backpropagation, Conjugate Gradient, and Early Stopping

**Rich Caruana**
CALD, CMU
5000 Forbes Ave.
Pittsburgh, PA 15213
*caruana@cs.cmu.edu*

**Steve Lawrence**
NEC Research Institute
4 Independence Way
Princeton, NJ 08540
*lawrence@research.nj.nec.com*

**Lee Giles**
Information Sciences
Penn State University
University Park, PA 16801
*giles@ist.psu.edu*

## Abstract

The conventional wisdom is that backprop nets with excess hidden units generalize poorly. We show that nets with excess capacity generalize well when trained with backprop and early stopping. Experiments suggest two reasons for this: 1) Overfitting can vary significantly in different regions of the model. Excess capacity allows better fit to regions of high non-linearity, and backprop often avoids overfitting the regions of low non-linearity. 2) Regardless of size, nets learn task subcomponents in similar sequence. Big nets pass through stages similar to those learned by smaller nets. Early stopping can stop training the large net when it generalizes comparably to a smaller net. We also show that conjugate gradient can yield worse generalization because it overfits regions of low non-linearity when learning to fit regions of high non-linearity.

## 1   Introduction

It is commonly believed that large multi-layer perceptrons (MLPs) generalize poorly: nets with too much capacity overfit the training data. Restricting net capacity prevents overfitting because the net has insufficient capacity to learn models that are too complex. This belief is consistent with a VC-dimension analysis of net capacity vs. generalization: the more free parameters in the net the larger the VC-dimension of the hypothesis space, and the less likely the training sample is large enough to select a (nearly) correct hypothesis [2].

Once it became feasible to train large nets on real problems, a number of MLP users noted that the overfitting they expected from nets with excess capacity did not occur. Large nets appeared to generalize as well as smaller nets — sometimes better. The earliest report of this that we are aware of is Martin and Pittman in 1991: *"We find only marginal and inconsistent indications that constraining net capacity improves generalization"* [7].

We present empirical results showing that MLPs with excess capacity often do not overfit. On the contrary, we observe that large nets often generalize better than small nets of sufficient capacity. Backprop appears to use excess capacity to better fit regions of high non-linearity, while still fitting regions of low non-linearity smoothly. (This desirable behavior can disappear if a fast training algorithm such as conjugate gradient is used instead of backprop.) Nets with excess capacity trained with backprop appear first to learn models similar to models learned by smaller nets. If early stopping is used, training of the large net can be halted when the large net's model is similar to models learned by smaller nets.

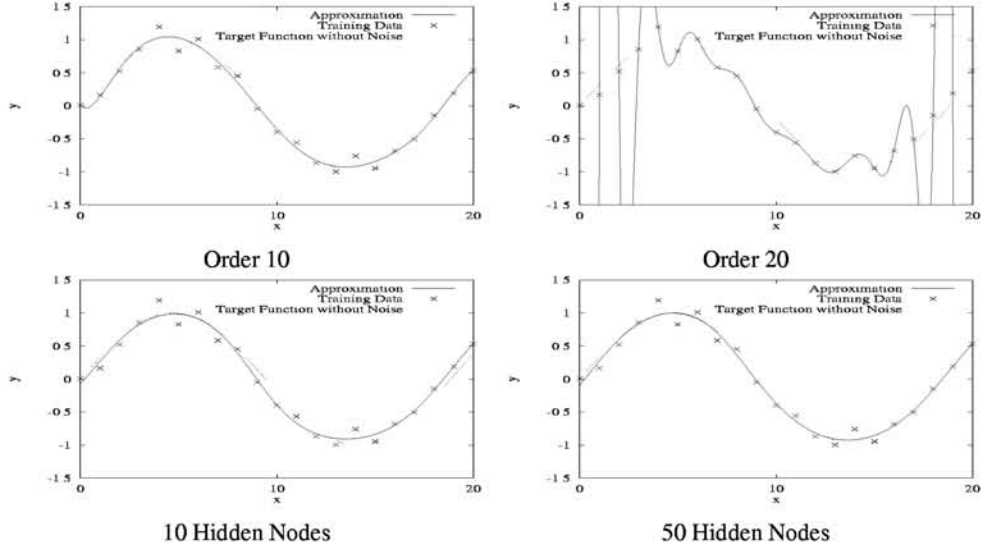

Order 10                           Order 20

10 Hidden Nodes                  50 Hidden Nodes

Figure 1: Top: Polynomial fit to data from $y = \sin(x/3) + \nu$. Order 20 overfits. Bottom: Small and large MLPs fit to same data. The large MLP does not overfit significantly more than the small MLP.

## 2 Overfitting

Much has been written about overfitting and the bias/variance tradeoff in neural nets and other machine learning models [2, 12, 4, 8, 5, 13, 6]. The top of Figure 1 illustrates polynomial overfitting. We created a training dataset by evaluating $y = \sin(x/3) + \nu$ at $0, 1, 2, \ldots, 20$ where $\nu$ is a uniformly distributed random variable between -0.25 and 0.25. We fit polynomial models with orders 2-20 to the data. Underfitting occurs with order 2. The fit is good with order 10. As the order (and number of parameters) increases, however, significant overfitting (poor generalization) occurs. At order 20, the polynomial fits the training data well, but interpolates poorly.

The bottom of Figure 1 shows MLPs fit to the data. We used a single hidden layer MLP, backpropagation (BP), and 100,000 stochastic updates. The learning rate was reduced linearly to zero from an initial rate of 0.5 (reducing the learning rate improves convergence, and linear reduction performs similarly to other schedules [3]). This schedule and number of updates trains the MLPs to completion. (We examine early stopping in Section 4.) As with polynomials, the smallest net with one hidden unit (HU) (4 weights weights) underfits the data. The fit is good with two HU (7 weights). Unlike polynomials, however, networks with 10 HU (31 weights) and 50 HU (151 weights) also yield good models. MLPs with seven times as many parameters as data points trained with BP do not significantly overfit this data. The experiments in Section 4 confirm that this bias of BP-trained MLPs towards smooth models is not limited to the simple 2-D problem used here.

## 3 Local Overfitting

Regularization methods such as weight decay typically assume that overfitting is a global phenomenon. But overfitting can vary significantly in different regions of a model. Figure 2 shows polynomial fits for data generated from the following equation:

$$y = \begin{cases} -\cos(x) + \nu & 0 \leq x < \pi \\ \cos(3(x - \pi)) + \nu & \pi \leq x \leq 2\pi \end{cases} \qquad \text{(Equation 1)}$$

Five equally spaced points were generated in the first region, and 15 in the second region, so that the two regions have different data densities and different underlying functions. Overfitting is different in the two regions. In Figure 2 the order 6 model fits the left region

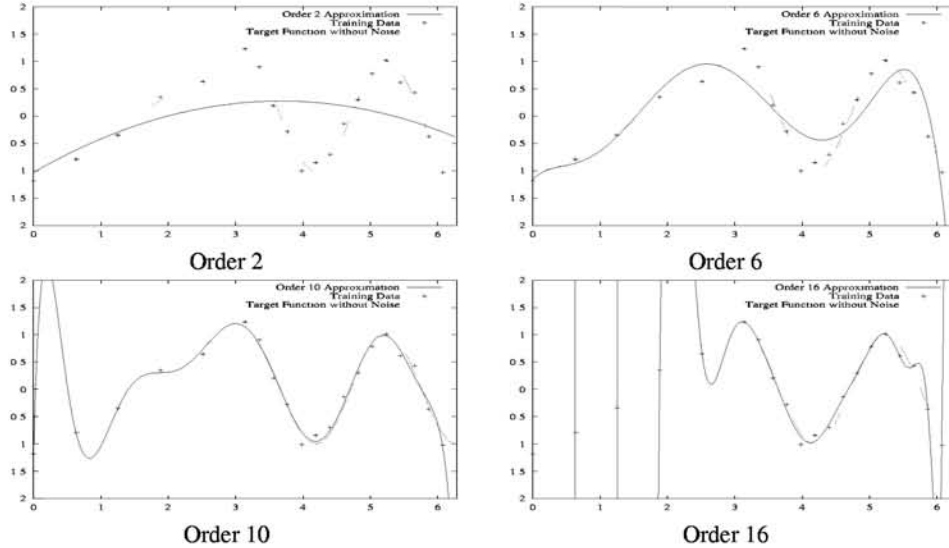

Figure 2: Polynomial approximation of data from Equation 1 as the order of the model is increased from 2 to 16. The overfitting behavior differs in the left and right hand regions.

well, but larger models overfit it. The order 6 model underfits the region on the right, and the order 10 model fits it better. No model performs well on both regions. Figure 3 shows MLPs trained on the same data (20,000 batch updates, learning rate linearly reduced to zero starting at 0.5). Small nets underfit. Larger nets, however, fit the entire function well *without significant overfitting in the left region.*

The ability of MLPs to fit both regions of low and high non-linearity well (without overfitting) depends on the training algorithm. Conjugate gradient (CG) is the most popular second order method. CG results in lower *training* error for this problem, but overfits significantly. Figure 4 shows results for 10 trials for BP and CG. Large BP nets generalize better on this problem — even the optimal size CG net is prone to overfitting. The degree of overfitting varies in different regions. When the net is large enough to fit the region of high non-linearity, overfitting is often seen in the region of low non-linearity.

## 4 Generalization, Network Capacity, and Early Stopping

The results in Sections 2 and 3 suggest that BP nets are less prone to overfitting than expected. But MLPs can and do overfit. This section examines overfitting vs. net size on seven problems: NETtalk [10], 7 and 12 bit parity, an inverse kinematic model for a robot arm (thanks to Sebastian Thrun for the simulator), Base 1 and Base 2: two sonar modeling problems using data collected from a robot wondering hallways at CMU, and vision data used to learn to steer an autonomous car [9]. These problems exhibit a variety of characteristics. Some are Boolean. Others are continuous. Some have noise. Others are noise-free. Some have many inputs or outputs. Others have few inputs or outputs.

### 4.1 Results

For each problem we used small training sets (100–1000 points, depending on the problem) so that overfitting was possible. We trained fully connected feedforward MLPs with one hidden layer whose size varied from 2 to 800 HU (about 500–100,000 parameters). All the nets were trained with BP using stochastic updates, learning rate 0.1, and momentum 0.9.

We used early stopping for regularization because it doesn't interfere with backprop's ability to control capacity locally. Early stopping combined with backprop is so effective that very large nets can be trained without significant overfitting. Section 4.2 explains why.

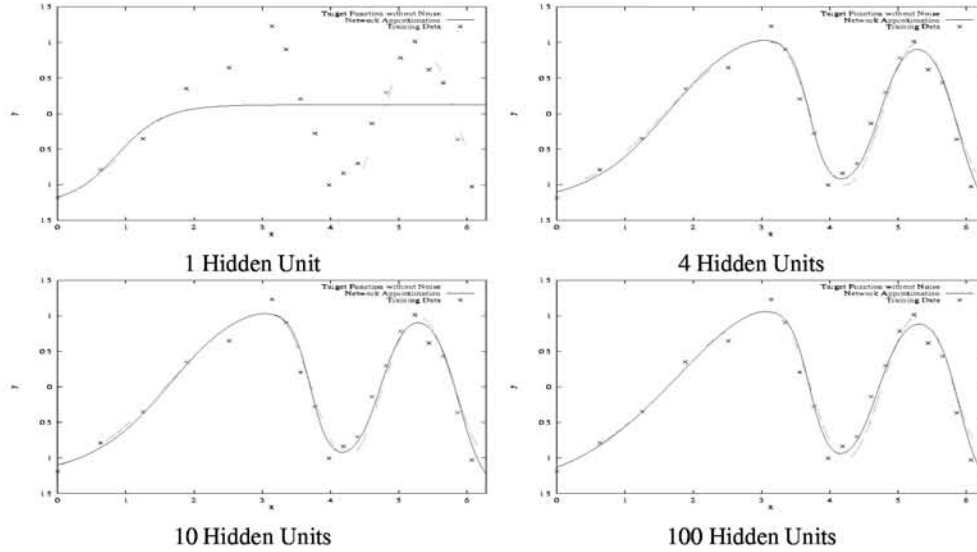

Figure 3: MLP approximation using backpropagation (BP) training of data from Equation 1 as the number of hidden units is increased. No significant overfitting can be seen.

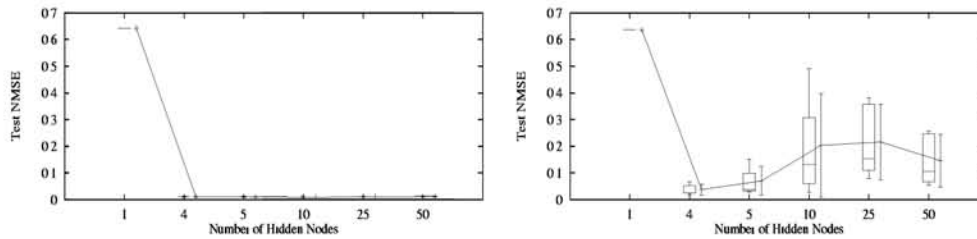

Figure 4: Test Normalized Mean Squared Error for MLPs trained with BP (left) and CG (right). Results are shown with both box-whiskers plots and the mean plus and minus one standard deviation.

Figure 5 shows generalization curves for four of the problems. Examining the results for all seven problems, we observe that on only three (Base 1, Base 2, and ALVINN), do nets that are too large yield worse generalization than smaller networks, but the loss is surprisingly small. Many trials were required before statistical tests confirmed that the differences between the optimal size net and the largest net were significant. Moreover, the results suggest that generalization is hurt more by using a net that is a little too small than by using one that is far too large, i.e., it is better to make nets too large than too small.

For most tasks and net sizes, we trained well beyond the point where generalization performance peaked. Because we had complete generalization curves, we noticed something unexpected. On some tasks, small nets overtrained considerably. The NETtalk graph in Figure 5 is a good example. Regularization (e.g., early stopping) is critical for nets of all sizes — not just ones that are *too big*. Nets with restricted capacity can overtrain.

### 4.2 Why Excess Capacity Does Not Hurt

BP nets initialized with small weights can develop large weights only after the number of updates is large. Thus BP nets consider hypotheses with small weights before hypotheses with large weights. Nets with large weights have more representational power, so simple hypotheses are explored before complex hypotheses.

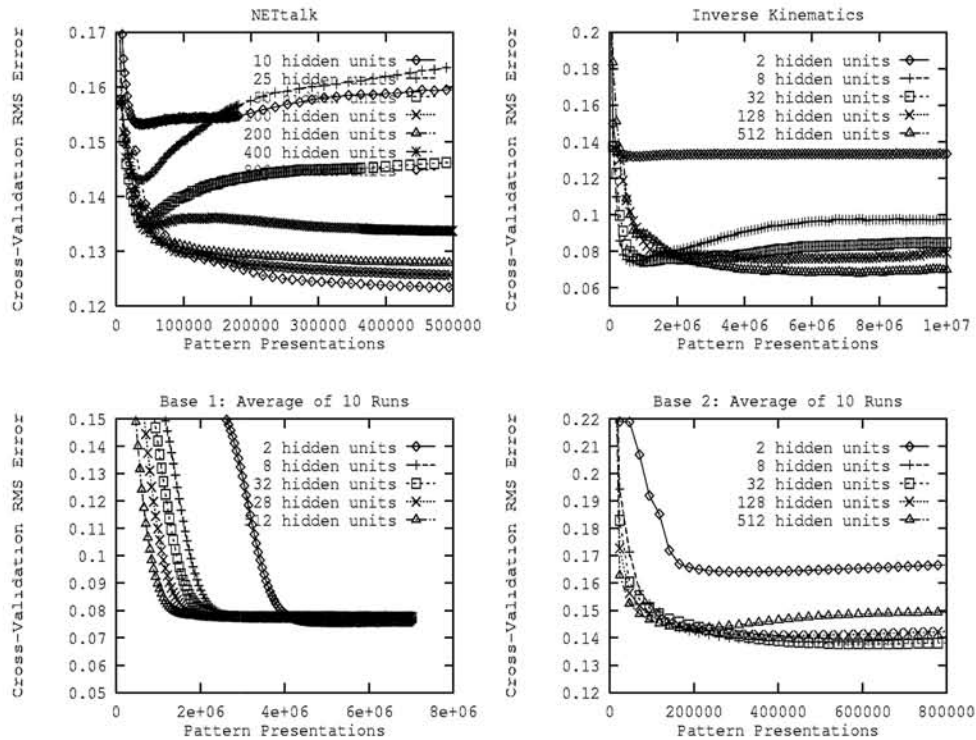

Figure 5: Generalization performance vs. net size for four of the seven test problems.

We analyzed *what* nets of different size learn *while* they are trained. We compared the input/output behavior of nets at different stages of learning on large samples of test patterns. We compare the input/output behavior of two nets by computing the squared error between the predictions made by the two nets. If two nets make the same predictions for all test cases, they have learned the same model (even though each model is represented differently), and the squared error between the two models is zero. If two nets make different predictions for test cases, they have learned different models, and the squared error between them is large. This is not the error the models make predicting the true labels, but the difference between predictions made by two different models. Two models can have poor generalization (large error on true labels), but have near zero error compared to each other if they are similar models. But two models with good generalization (low error on true labels) must have low error compared to each other.

The first graph in Figure 5 shows learning curves for nets with 10, 25, 50, 100, 200, and 400 HU trained on NETtalk. For each size, we saved the net from the epoch that generalized best on a large test set. This gives us the best model of each size found by backprop. We then trained a BP net with 800 HU, and after each epoch compared this net's model with the best models saved for nets of 10–400 HU. This lets us compare the sequence of models learned by the 800 HU net to the best models learned by smaller nets.

Figure 6 shows this comparison. The horizontal axis is the number of backprop passes applied to the 800 HU net. The vertical axis is the error between the 800 HU net model and the best model for each smaller net. The 800 HU net starts off distant from the good smaller models, then becomes similar to the good models, and then diverges from them. This is expected. What is interesting is that the 800 HU net first becomes closest to the best

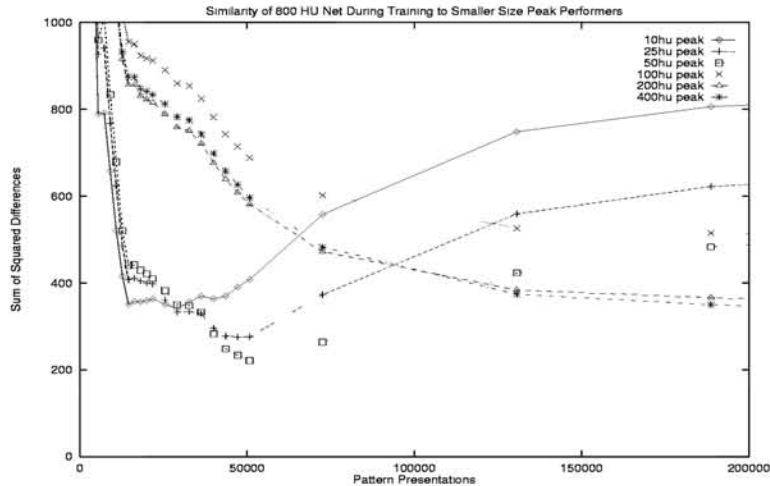

Figure 6: I/O similarity during training between an 800 hidden unit net and smaller nets (10, 25, 50, 100, 200, and 400 hidden units) trained on NETtalk.

10 HU net, then closest to the 25 HU net, then closest to the 50 HU net, etc. As it is trained, the 800 HU net learns a sequence of models similar to the models learned by smaller nets. If early stopping is used, training of the 800 HU net can be stopped when it behaves similar to the best model that could be learned with nets of 10, 25, 50, ... HU. Large BP nets learn models similar to those learned by smaller nets. *If a BP net with too much capacity would overfit, early stopping could stop training when the model was similar to a model that would have been learned by a smaller net of optimal size.*

The error *between* models is about 200–400, yet the *generalization* error is about 1600. The models are much closer to each other than any of them are to the true model. With early stopping, what counts is the closest approach of each model to the target function, not where models end up late in training. With early stopping there is little disadvantage to using models that are too large because their learning trajectories are similar to those followed by smaller nets of more optimal size.

## 5   Related Work

Our results show that models learned by backprop are biased towards "smooth" solutions. As nets with excess capacity are trained, they first explore smoother models similar to the models smaller nets would have learned. Weigend [11] performed an experiment that showed BP nets learn a problem's eigenvectors in sequence, learning the 1st eigenvector first, then the 2nd, etc. His result complements our analysis of what nets of different sizes learn: if large nets learn an eigenvector sequence similar to smaller nets, then the models learned by the large net will pass through intermediate stages similar to what is learned by small nets (but iff nets of different sizes learn the eigenvectors equally well, which is an assumption we do not need to make.)

Theoretical work by [1] supports our results. Bartlett notes: *"the VC-bounds seem loose; neural nets often perform successfully with training sets that are considerably smaller than the number of weights."* Bartlett shows (for classification) that the number of training samples only needs to grow according to $A^{2l}$ (ignoring log factors) to avoid overfitting, where $A$ is a bound on the total weight magnitudes and $l$ is the number of layers in the network. This result suggests that a net with smaller weights will generalize better than a similar net with large weights. Examining the weights from BP and CG nets shows that BP training typically results in smaller weights.

# 6  Summary

Nets of *all* sizes overfit some problems. But generalization is surprisingly insensitive to excess capacity if the net is trained with backprop. Because BP nets with excess capacity learn a sequence of models functionally similar to what smaller nets learn, early stopping can often be used to stop training large nets when they have learned models similar to those learned by smaller nets of optimal size. This means there is little loss in generalization performance for nets with excess capacity if early stopping can be used.

Overfitting is not a global phenomenon, although methods for controlling it often assume that it is. Overfitting can vary significantly in different regions of the model. MLPs trained with BP use excess parameters to improve fit in regions of high non-linearity, while not significantly overfitting other regions. Nets trained with conjugate gradient, however, are more sensitive to net size. BP nets appear to be better than CG nets at avoiding overfitting in regions with different degrees of non-linearity, perhaps because CG is more effective at learning more complex functions that overfit training data, while BP is biased toward learning smoother functions.

# References

[1] Peter L. Bartlett. For valid generalization the size of the weights is more important than the size of the network. In *Advances in Neural Information Processing Systems*, volume 9, page 134. The MIT Press, 1997.

[2] E.B. Baum and D. Haussler. What size net gives valid generalization? *Neural Computation*, 1(1):151–160, 1989.

[3] C. Darken and J.E. Moody. Note on learning rate schedules for stochastic optimization. In *Advances in Neural Information Processing Systems*, volume 3, pages 832–838. Morgan Kaufmann, 1991.

[4] S. Geman et al. Neural networks and the bias/variance dilemma. *Neural Computation*, 4(1):1–58, 1992.

[5] A. Krogh and J.A. Hertz. A simple weight decay can improve generalization. In *Advances in Neural Information Processing Systems*, volume 4, pages 950–957. Morgan Kaufmann, 1992.

[6] Y. Le Cun, J.S. Denker, and S.A. Solla. Optimal Brain Damage. In D.S. Touretzky, editor, *Advances in Neural Information Processing Systems*, volume 2, pages 598–605, San Mateo, 1990. (Denver 1989), Morgan Kaufmann.

[7] G.L. Martin and J.A. Pittman. Recognizing hand-printed letters and digits using backpropagation learning. *Neural Computation*, 3:258–267, 1991.

[8] J.E. Moody. The effective number of parameters: An analysis of generalization and regularization in nonlinear learning systems. In *Advances in Neural Information Processing Systems*, volume 4, pages 847–854. Morgan Kaufmann, 1992.

[9] D.A. Pomerleau. Alvinn: An autonomous land vehicle in a neural network. In D.S. Touretzky, editor, *Advances in Neural Information Processing Systems*, volume 1, pages 305–313, San Mateo, 1989. (Denver 1988), Morgan Kaufmann.

[10] T. Sejnowski and C. Rosenberg. Parallel networks that learn to pronounce English text. *Complex Systems*, 1:145–168, 1987.

[11] A. Weigend. On overfitting and the effective number of hidden units. In *Proceedings of the 1993 Connectionist Models Summer School*, pages 335–342. Lawrence Erlbaum Associates, 1993.

[12] A.S. Weigend, D.E. Rumelhart, and B.A. Huberman. Generalization by weight-elimination with application to forecasting. In *Advances in Neural Information Processing Systems*, volume 3, pages 875–882. Morgan Kaufmann, 1991.

[13] D. Wolpert. On bias plus variance. *Neural Computation*, 9(6):1211–1243, 1997.
